# Minimising Contrastive Divergence in Noisy, Mixed-mode VLSI Neurons

**Hsin Chen, Patrice Fleury and Alan F. Murray**
School of Engineering and Electronics
Edinburgh University
Mayfield Rd., Edinburgh
EH9 3JL, UK
{*hc, pcdf, afm*}*@ee.ed.ac.uk*

## Abstract

This paper presents VLSI circuits with continuous-valued probabilistic behaviour realized by injecting noise into each computing unit(neuron). Interconnecting the noisy neurons forms a Continuous Restricted Boltzmann Machine (CRBM), which has shown promising performance in modelling and classifying noisy biomedical data. The Minimising-Contrastive-Divergence learning algorithm for CRBM is also implemented in mixed-mode VLSI, to adapt the noisy neurons' parameters on-chip.

## 1  Introduction

As interests in interfacing electronic circuits to biological cells grows, an intelligent embedded system able to classify noisy and drifting biomedical signals becomes important to extract useful information at the bio-electrical interface. Probabilistic neural computation utilises probability to generalise the natural variability of data, and is thus a potential candidate for underpinning such intelligent systems. To date, probabilistic computation has been unable to deal with the continuous-valued nature of biomedical data, while remaining amenable to hardware implementation. The Continuous Restricted Boltzmann Machine(CRBM) has been shown to be promising in the modelling of noisy and drifting biomedical data[1][2], with a simple Minimising-Contrastive-Divergence(MCD) learning algorithm[1][3]. The CRBM consists of continuous-valued stochastic neurons that adapt their "internal noise" to code the variation of continuous-valued data, dramatically enriching the CRBM's representational power. Following a brief introduction of the CRBM, the VLSI implementation of the noisy neuron and the MCD learning rule are presented.

## 2  Continuous Restricted Boltzmann Machine

Let $s_i$ represent the state of neuron $i$, and $w_{ij}$ the connection between neuron $i$ and neuron $j$. A noisy neuron $j$ in the CRBM has the following form:

$$s_j = \varphi_j \left( \sum_i w_{ij} s_i + \sigma \cdot N_j(0,1) \right), \qquad (1)$$

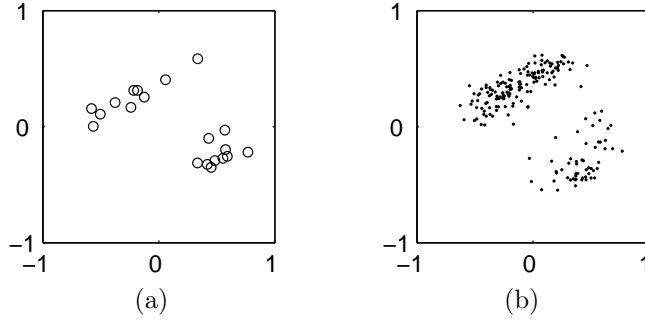

Figure 1: (a)20 two-dimensional artificial training data (b)20-step reconstruction by the CRBM after 30,000 epochs' fixed-step training

$$\text{with} \quad \varphi_j(x_j) = \theta_L + (\theta_H - \theta_L) \cdot \frac{1}{1 + \exp(-a_j x_j)} \tag{2}$$

where $N_j(0,1)$ refers to a unit Gaussian noise with zero mean, $\sigma$ a noise-scaling constant, and $\varphi_j(\cdot)$ the sigmoid function with asymptotes at $\theta_H$ and $\theta_L$. Parameter $a_j$ is the "noise-control factor", controlling the neuron's output nonlinearity such that a neuron $j$ can learn to become near-deterministic (small $a_j$), continuous-stochastic (moderate $a_j$), or binary-stochastic (large $a_j$)[4][1].

A CRBM consists of one visible and one hidden layer of noisy neurons with inter-layer connections defined by a weight matrix $\{\mathbf{W}\}$. By minimizing the "Contrastive Divergence" between the training data and the one-step Gibbs sampled data [3], the parameters $\{w_{ij}\}$ and $\{a_j\}$ evolve according to the following equations [1]

$$\Delta \hat{w}_{ij} = \eta_w (\langle s_i s_j \rangle - \langle \hat{s}_i \hat{s}_j \rangle) \tag{3}$$

$$\Delta \hat{a}_j = \frac{\eta_a}{a_j^2} \left( \langle s_j^2 \rangle - \langle \hat{s}_j^2 \rangle \right) \tag{4}$$

where $\hat{s}_i$ and $\hat{s}_j$ denote the one-step sampled state of neuron $i$ and $j$ respectively, and $\langle \cdot \rangle$ refers to the expectation over all training data. $\eta_w$ and $\eta_a$ denote the learning rates for parameters $\{w_{ij}\}$ and $\{a_j\}$, respectively. Following [5], Eq.(3)and(4) are further simplified to fixed-step directional learning, rather than variable accurate-step learning, as following.

$$\Delta \hat{w}_{ij} = \eta_w sign \left( \langle s_i s_j \rangle_4 - \langle \hat{s}_i \hat{s}_j \rangle_4 \right) \tag{5}$$

$$\Delta \hat{a}_j = \eta_a sign \left( \langle s_j^2 \rangle_4 - \langle \hat{s}_j^2 \rangle_4 \right) \tag{6}$$

Note that the denominator $1/a_j^2$ in Eq.(4) is also absorbed and $\langle \cdot \rangle_4$ indicates that the expectation operator will be approximated by the average of *four* data as opposed to all training data. To validate the simplification above, a CRBM with 2 visible neurons and 4 hidden neurons was trained to model the two-dimensional data distribution defined by 20 training data (Fig.1a), with $\eta_w = 1.5$, $\eta_a = 15$ for visible neurons, and $\eta_a = 1$ for hidden neurons [1]. After 30,000 training updates, the trained CRBM reconstructed the same data distribution (Fig.1b) from 200 initially random-distributed data, indicating that the simplification above reduces the hardware complexity at the cost of a slightly slower convergence time.

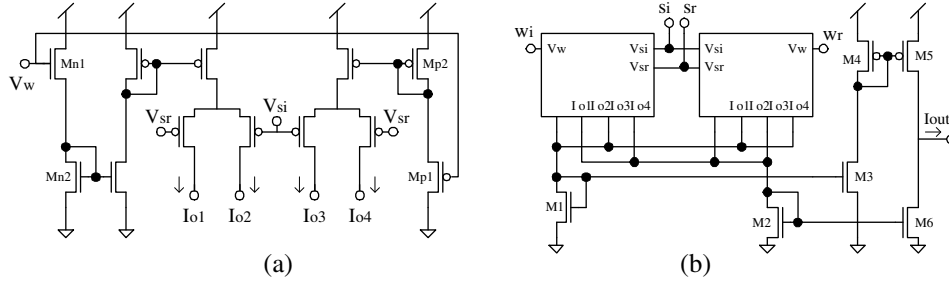

Figure 2: The circuits of the four-quadrant multiplier (a)one computing cell (b)full circuit composed of two computing cell

## 3 Noisy neuron with variable nonlinearity

The circuits were fabricated on the AMS $0.6\mu$m 2P3M CMOS process, which allows a power supply voltage of five volts. Therefore, the states of neurons $\{s_i\}$ and the corresponding weights $\{w_{ij}\}$ are designed to be represented by voltage in [1.5, 3.5]V and [0,5]V respectively, with both arithmetical zeros at 2.5V. As both $s_i$ and $w_{ij}$ are real numbers, a four-quadrant multiplier is required to calculate $w_{ij}s_i$

### 3.1 Four-quadrant multiplier

While the Chible four-quadrant multiplier [6] has a simple architecture with a wide input range, the reference zero of one of its inputs is process-dependent. Though only relative values of weights matter for the neurons, the process-dependent reference becomes nontrivial if the same four-quadrant multiplier is used to implement the MCD learning rule. We thus proposed a 'modified Chible multiplier' composed of two computing cells, as shown in Fig.2, to allow external control of reference zeros of both inputs.

Each computing cell contains two differential pairs biased by two complementary branches, Mn1-Mn2 and Mp1-Mp2. $(I_{o1}-I_{o2})$ is thus proportional to $(V_w-V_{th,n1}-nV_{th,n2})(V_{si}-V_{sr})$ when $V_w > (V_{th,n1}+nV_{th,n2})$ [2], and $(I_{o3}-I_{o4})$ proportional to $(n^2Vdd-V_w-V_{th,p1}-nV_{th,p2})(V_{sr}-V_{si})$ when $V_w < (n^2Vdd-V_{th,p1}-nV_{th,p2})$[6]. Subject to careful design of the complementary biasing transistors[6], $(V_{th,n1}+nV_{th,n2}) \approx (n^2Vdd-V_{th,p1}-nV_{th,p2}) \approx Vdd/2$. Combining the two differential currents then gives

$$I_o = (I_{o1}+I_{o3})-(I_{o2}+I_{o4}) = I(V_w)\cdot(V_{si}-V_{sr}) \tag{7}$$

With $w_i$ input to one computing cell and $w_r$ to the other cell, as shown in Fig.2b, M1-M6 generates an output current $I_{out} \propto (w_i-w_r)(s_i-s_r)$. The measured DC characteristic from a fabricated chip is shown in Fig.4(a)

### 3.2 Noisy neuron

Fig.3 shows the circuit diagram of a noisy neuron. The four-quadrant multipliers output a total current proportional to $\sum_i w_{ij}s_i$, while the differential pair, Mna and

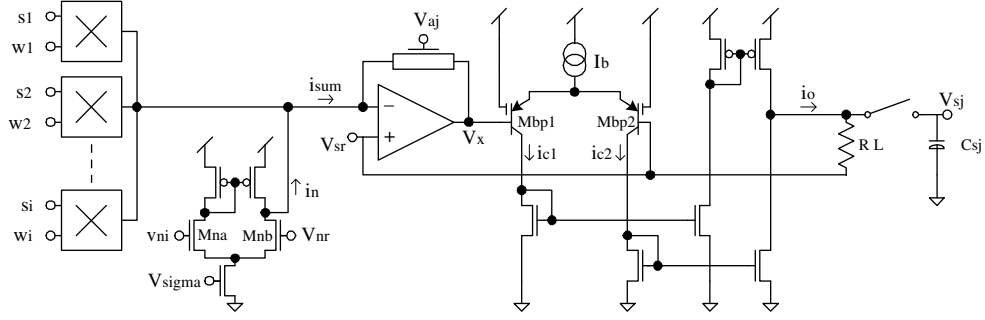

Figure 3: The circuit diagram of a noisy neuron

Mnb, transforms noise voltage $v_{ni}$ into a noise current $i_n = g_m(v_{ni} - V_{nr})$, where $V_{sigma}$ controls the transconductance $g_m$ and thus scales the noise current as $\sigma$ in Eq.(1). The current-to-voltage converter, composed of an operational amplifier and an voltage-controlled active resistor[7], then sums all currents, outputting a voltage $V_x = V_{sr} - i_{sum} \cdot R(V_{aj})$ to the sigmoid function.

The exponential nonlinearity of the sigmoid function is achieved by operating the PMOS differential pair, Mbp1-Mbp2, in the lateral-bipolar mode [8], resulting in a differential output current as following

$$i_o = i_{c1} - i_{c2} = I_b \cdot \phi(\frac{I_{sum} \cdot R(V_{aj})}{V_t})$$ (8)

where $\phi(\cdot)$ denotes the $\varphi(\cdot)$ with $\theta_H = -\theta_L = 1$, and $V_t = kT/q$ is the thermal voltage. The resistor $R_L$ finally converts $i_o$ into a output voltage $v_o = i_o R_L + V_{sr}$. Eq.(8) implies that $V_{aj}$ controls the feedback resistance of the I-V converter, and consequently adapts the nonlinearity of the sigmoid function (which appears as $a_j$ in Eq.(1)). With various $V_{aj}$, the measured DC characteristic (chip result) of the sigmoid function is shown in Fig.4b.

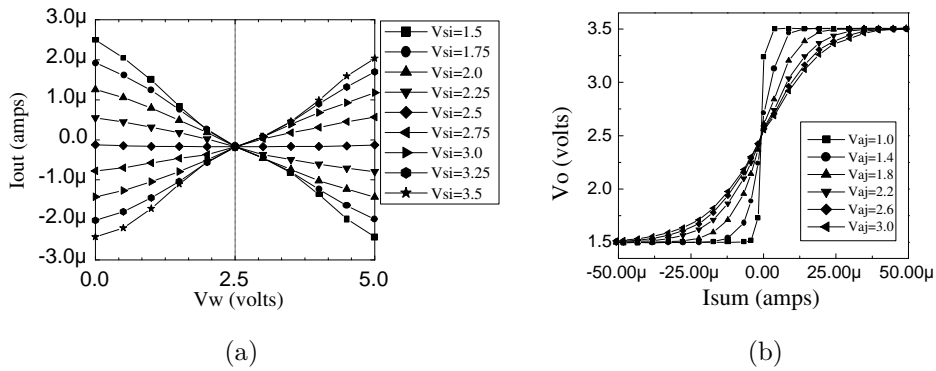

(a)                                           (b)

Figure 4: The measured DC characteristics of (a) four-quadrant multiplier (b)sigmoid function with variable nonlinearity controlled by $V_{aj}$

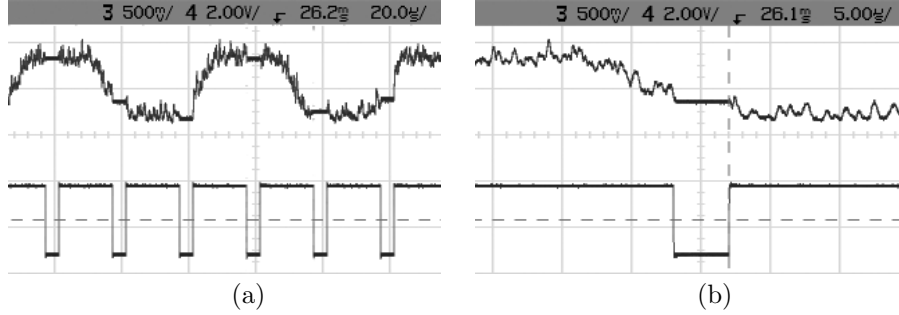

(a)                                                      (b)

Figure 5: (a)The measured output of a noisy neuron (upper trace) and the switching
signal (lower trace) that samples $V_{sj}$ (b) Zooming-in of the second sample in(a)

Fig.5 shows the measured output of a noisy neuron (upper trace) with $\{s_i\}$ sweeping
between 1.5 and 3.5V, $\{w_i\}$=4V, $V_{aj}$=1.8V, and $v_{ni}$ generated by LFSR (Linear
Feedback Shift Register) [9] with an amplitude of 0.4V. The $\{s_i\}$ and $\{w_i\}$ above
forced the neuron's output to sweep a sigmoid-shaped curve as Fig.4b, while the
input noise disturbed the curve to achieve continous-valued probabilistic output. A
neuron state $V_{sj}$ was sampled periodically and held with negligible clock feedthrough
whenever the switch opened(went low).

## 4  Minimising-Contrastive-Divergence learning on chip

The MCD learning for the Product of Experts[3] has been successfully implemented
and reported in [10]. The MCD learning for CRBM is therefore implemented simply
by replacing the following two circuits. First, the four-quadrant multiplier described
in Sec.3.1 is substituted for the two-quadrant multiplier in [10] to enhance learning
flexibility; secondly, a pulse-coded learning circuit, rather than the analogue weight-
changing circuit in [10], is employed to allow not only accurate learning steps but
also refresh of dynamically-held parameters.

### 4.1  MCD learning for CRBM

Fig.6 shows the block diagram of the VLSI implementation of the MCD learning
rules for the noisy neurons, along with the digital control signals. In learning mode
(LER/$\overline{\text{REF}}$=1), the initial states $s_i$ and $s_j$ are first sampled by clock signals $CK_{si}$
and $CK_{sj}$, resulting in a current $I_+$ at the output of four-quadrant multiplier.
After $CK_+$ samples and holds $I_+$, the one-step reconstructed states $\hat{s}_i$ and $\hat{s}_j$ are
sampled by $CK_{sip}$ and $CK_{sjp}$ to produce another current $I_-$. $CK_q$ then samples
and holds the output of the current subtracter $I_{sub}$, which represents the difference
between initial data and one-step Gibbs sampled data. Repeating the above clocking
sequence for *four* cycles, four $I_{sub}$ are accumulated and averaged to derive $I_{ave}$,
representing $\langle s_i s_j \rangle_4 - \langle \hat{s}_i \hat{s}_j \rangle_4$ in equation(5). Finally, $I_{ave}$ is compared to a reference
current to determine the learning direction DIR, and the learning circuit, triggered
by $CK_{up}$, updates the parameter once. The dash-lined box represents the voltage-
limiting circuit used only for parameter $\{a_j\}$, whose voltage range should be limited
to ensure normal operation of the voltage-controlled active resistor in Fig.3. In
refresh mode (LER/$\overline{\text{REF}}$=1), the signal REFR rather than DIR determines the
updating direction, maintaining the weight to a reference value.

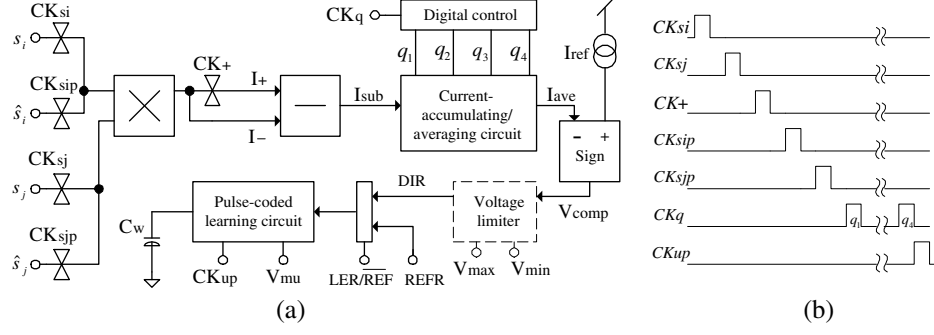

(a)

(b)

Figure 6: (a)The block diagram of VLSI implementation of MCD learning rules described in Eq.(5)(6) (b)The digital control signals

The subtracter, accumulator and current comparator in Fig.6 are dominated by the dynamic current mirror[11] and are the same as those used in [10]. The following subsections therefore focus on the pulse-coded learning circuit and the measurement results of on-chip MCD learning.

## 4.2   The pulse-coded learning circuit

The pulse-coded learning circuit consists of a pulse generator (Fig.7a) and the learning cell proposed in [12] (Fig.7b). The stepsize of the learning cell is adjustable through $V_P$ and $V_N$ in Fig.7b [12]. However, transistor nonlinearities and process variations do not allow different and accurate learning rates to be set for various parameters in the same chip ($\{a_j\}$ and $\{w_{ij}\}$ in our case). We therefore apply a width-variable pulse to the enabling input (EN) of the learning cell, controlling the learning step precisely by monitoring the pulse width off-chip. As the input capacitance of each learning cell is less than 0.1pF, one pulse generator can control all the learning cells with the same learning rate. The simulation in Sec.2 implies that only three pulse generators are required for $\eta_w$, $\eta_{av}$, and $\eta_{ah}$. The pulse generator is therefore a simple way to achieve accurate control.

The pulse generator is largely a D-type flip-flop whose output $V_{pulse}$ is initially reset to low via $\overline{reset}$. $V_{pulse}$ then goes high on the rising edge of $CK_{up}$, while the

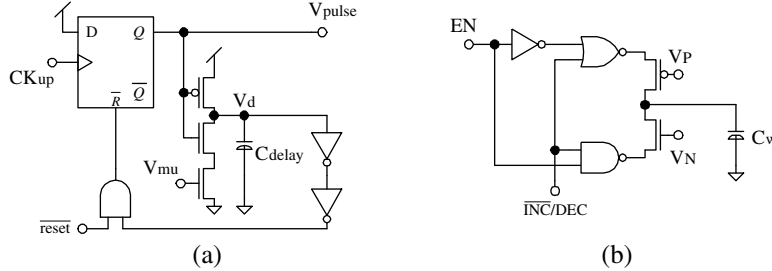

(a)                                                (b)

Figure 7: The pulse-coded learning circuit composed of (a)a pulse generator and (b)a learning cell proposed in [12]

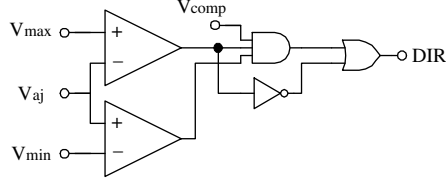

Figure 8: The voltage-limiting circuit

capacitor $C_{delay}$ prevents $V_d$ from going from high to low instantly. Eventually, $V_{pulse}$ is reset to zero as soon as $V_d$ is discharged. During the positive pulse, the learning cell charges or discharges the voltage stored on $C_w$[12], according to the directional input $\overline{INC}$/DEC. Varying $V_{mu}$ controls the pulse width accurately from 10ns ($V_\eta = 2.5V$) to 5us ($V_\eta = 0.9V$), amounting to learning stepsize from 1mV to 500mV as $V_N = 0.75V$, $V_P = 4.29V$, and $C_w = 1pF$.

### 4.3 Voltage-limiting circuit

Although Eq.(6) indicates that $\{a_j\}$ can be adapted with the same learning circuit simply by substituting $s_j$ and $\hat{s}_j$ for $s_i$ and $\hat{s}_i$ in Fig.6, the voltage $V_{aj}$ should be confined in [1,3]V, to ensure normal operation of the voltage-controlled active resistor in Fig.3. A voltage-limiting circuit as shown in Fig.8 is thus designed to limit the range of $V_{aj}$, defined by $V_{max}$ and $V_{min}$ through two voltage comparators. As $V_{max} > V_{aj} > V_{mi}$, DIR equals $V_{comp}$, i.e. the MCD learning rule decides the learning direction. However, DIR goes high to enforce decreasing $V_{aj}$ when $V_{aj} > V_{max} > V_{min}$, while DIR goes low to enforce increasing $V_{aj}$ when $V_{max} > V_{min} > V_{aj}$.

### 4.4 On-chip learning

Two MCD learning circuits, one for $\{w_{ij}\}$ and the other for $\{a_j\}$, have been fabricated successfully. Fig.9 shows the measured on-chip learning of both parameters with (a) different learning rates (b) different learning directions. To ease testing, $s_i$ and $\hat{s}_i$ are fixed at 3.5V, while $s_j$ and $\hat{s}_j$ alternate between 1.5V and 3.5V, as shown by the traces SJ and SJ_P in Fig.9. With the reference zero being defined at

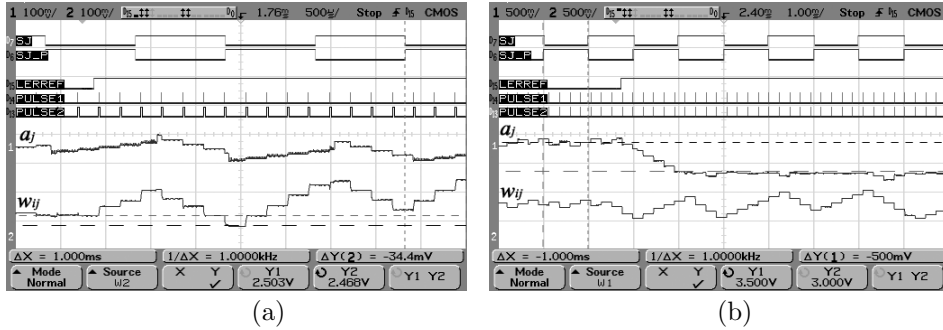

(a)                                        (b)

Figure 9: Measurement of parameter $a_j$ and $w_{ij}$ learning in (a)different learning rates (b)different directions

2.5V, the parameters should learn down when $s_j$=3.5V and $\hat{s_j}$=1.5V, and learn up when $s_j$=1.5V and $\hat{s_j}$=3.5V.

In Fig.9a, both parameters were initially refreshed to 2.5V when signal LERREF is low, and subsequently started to learn up and down in response to the changing SJ and SJ_P as LERREF goes high. As controlled by different pulse widths (PULSE1 and PULSE2), the two parameters were updated with different stepsizes (10mV and 34mV) but in the same direction. The trace of parameter $a_j$ shows digital noise attributable to sub-optimal layout, and has been improved in a subsequent design. In Fig.9b, both parameters were refreshed to 3.5V, a voltage higher than $V_{max}$=3V set for $a_j$. Therefore, the learning circuit forces $a_j$ to decrease toward $V_{max}$, while $w_{ij}$ remains learning up and down as Fig.9a.

## 5   Conclusion

Fabricated CMOS circuits have been presented and the implemention of noisy neural computation that underlies the CRBM has been demonstrated. The promising measured results show that the CRBM is, as has been inferred in the past[1], amenable to mixed-mode VLSI. This makes possible a VLSI system with continuous-valued probabilistic behaviour and on-chip adaptability, adapting its "internal noise" to model the "external noise" in its environment. A full CRBM system with two visible and four hidden neurons has thus been implemented to examine this concept. The neurons in the proof-of-concept CRBM system are hard-wired to each other and the multi-channel uncorrelated noise sources implemented by the LFSR [9]. A scalable design will thus be an essential next step before pratical biomedical applications. Furthermore, the CRBM system may open the possibility of utilising VLSI intrinsic noise for computation in the deep-sub-miron era.

## Footnotes

[1]constants $\theta_H = -\theta_L = 1$ and $\sigma = 0.2$ for all neurons

[2]$n$ is the slope factor of MOS transistor, and $V_{th,x}$ refers to the absolute value of transistor Mx's threshold voltage.

## References

[1] H. Chen and A. Murray, "A continuous restricted boltzmann machine with an implementable training algorithm," *IEE Proc. of Vision, Image and Signal Processing*, vol. 150, no. 3, pp. 153–158, 2003.

[2] T. Tang, H. Chen, and A. Murray, "Adaptive Stochastic Classifier for Noisy pH-ISFET Measurements," in *Proceedings of Thirteenth International Conference on Artificial Neural Networks (ICANN2003)*, (Istanbul, Turkey), pp. 638–645, Jun. 2003.

[3] G. E. Hinton, "Training products of experts by minimizing contrastive divergence," *Neural Computation*, vol. 14, no. 8, pp. 1771–1800, 2002.

[4] B. J. Frey, "Continuous sigmoidal belief networks trained using slice sampling," *Advances in Neural Information Processing Systems*, vol. 9, pp. 452–458, 1997.

[5] A. F. Murray, "Novelty detection using products of simple experts-a potential architecture for embedded systems," *Neural Networks*, vol. 14, no. 9, pp. 1257–1264, 2001.

[6] H. Chible, "Analog circuit for synapse neural networks vlsi implementation," *The 7th IEEE Int. Conf. on Electronics, Circuits and Systems (ICECS 2000)*, vol. 2, pp. 1004–1007, 2000.

[7] M. Banu and Y. Tsividis, "Floating voltage-controlled resistors in cmos technology," *Electronics Letters*, vol. 18, pp. 678–679, 1982.

[8] E. Vittoz, "Mos transistors operated in the lateral bipolar mode and their application in cmos technology," *IEEE Journal of Solid-State Circuits*, vol. sc-18, no. 3, pp. 273–279, 1983.

[9] J. Alspector, J. W. Gannett, S. Haber, M. B. Parker, and R. Chu, "A vlsi-efficient technique for generating multiple uncorrelated noise sources and its application to stochastic neural networks," *IEEE Trans. Circuits and Systems*, vol. 38, no. 1, pp. 109–123, 1991.

[10] P. Fleury and A. Murray, "Mixed-signal vlsi implementation of the product of experts' minimizing contrastive divergence learning scheme," in *IEEE Proc. of the Int. Sym. on Circuits and Systems (ISCAS 2003)*, vol. 5, (Bangkok, Thailand), pp. 653–656, May 2003.

[11] G. Wegmann and E. Vittoz, "Basic principles of accurate dynamic current mirrors," *IEE Proc. on Circuits, Devices and Systems*, vol. 137, pp. 95–100, April 1990.

[12] G. Cauwenberghs, "An analog vlsi recurrent neural network," *IEEE Tran. on Neural Networks*, vol. 7, pp. 346–360, Mar. 1996.
